# Bayesian Reconstruction of 3D Human Motion from Single-Camera Video

**Nicholas R. Howe**
Department of Computer Science
Cornell University
Ithaca, NY 14850
*nihowe@cs.cornell.edu*

**Michael E. Leventon**
Artificial Intelligence Lab
Massachusetts Institute of Technology
Cambridge, MA 02139
*leventon@ai.mit.edu*

**William T. Freeman**
MERL – a Mitsubishi Electric Research Lab
201 Broadway
Cambridge, MA 02139
*freeman@merl.com*

## Abstract

The three-dimensional motion of humans is underdetermined when the observation is limited to a single camera, due to the inherent 3D ambiguity of 2D video. We present a system that reconstructs the 3D motion of human subjects from single-camera video, relying on prior knowledge about human motion, learned from training data, to resolve those ambiguities. After initialization in 2D, the tracking and 3D reconstruction is automatic; we show results for several video sequences. The results show the power of treating 3D body tracking as an inference problem.

## 1 Introduction

We seek to capture the 3D motions of humans from video sequences. The potential applications are broad, including industrial computer graphics, virtual reality, and improved human-computer interaction. Recent research attention has focused on unencumbered tracking techniques that don't require attaching markers to the subject's body [4, 5], see [12] for a survey. Typically, these methods require simultaneous views from multiple cameras.

Motion capture from a single camera is important for several reasons. First, though underdetermined, it is a problem people can solve easily, as anyone viewing a dancer in a movie can confirm. Single camera shots are the most convenient to obtain, and, of course, apply to the world's film and video archives. It is an appealing computer vision problem that emphasizes inference as much as measurement.

This problem has received less attention than motion capture from multiple cameras. Goncalves et.al. rely on perspective effects to track only a single arm, and thus need not deal with complicated models, shadows, or self-occlusion [7]. Bregler & Malik develop a body tracking system that may apply to a single camera, but performance in that domain is

not clear; most of the examples use multiple cameras [4]. Wachter & Nagel use an iterated extended Kalman filter, although their body model is limited in degrees of freedom [12]. Brand [3] uses an learning-based approach, although with representational expressiveness restricted by the number of HMM states. An earlier version of the work reported here [10] required manual intervention for the 2D tracking.

This paper presents our system for single-camera motion capture, a learning-based approach, relying on prior information learned from a labeled training set. The system tracks joints and body parts as they move in the 2D video, then combines the tracking information with the prior model of human motion to form a best estimate of the body's motion in 3D. Our reconstruction method can work with incomplete information, because the prior model allows spurious and distracting information to be discarded. The 3D estimate provides feedback to influence the 2D tracking process to favor more likely poses.

The 2D tracking and 3D reconstruction modules are discussed in Sections 3 and 4, respectively. Section 4 describes the system operation and presents performance results. Finally, Section 5 concludes with possible improvements.

## 2   2D Tracking

The 2D tracker processes a video stream to determine the motion of body parts in the image plane over time. The tracking algorithm used is based on one presented by Ju et. al. [9], and performs a task similar to one described by Morris & Rehg [11]. Fourteen body parts are modeled as planar patches, whose positions are controlled by 34 parameters. Tracking consists of optimizing the parameter values in each frame so as to minimize the mismatch between the image data and a projection of the body part maps. The 2D parameter values for the first frame must be initialized by hand, by overlaying a model onto the 2D image of the first frame.

We extend Ju et. al.'s tracking algorithm in several ways. We track the entire body, and build a model of each body part that is a weighted average of several preceding frames, not just the most recent one. This helps eliminate tracking errors due to momentary glitches that last for a frame or two.

We account for self-occlusions through the use of support maps [4, 1]. It is essential to address this problem, as limbs and other body parts will often partly or wholly obscure one another. For the single-camera case, there are no alternate views to be relied upon when a body part cannot be seen.

The 2D tracker returns the coordinates of each limb in each successive frame. These in turn yield the positions of joints and other control points needed to perform 3D reconstruction.

## 3   3D Reconstruction

3D reconstruction from 2D tracking data is underdetermined. At each frame, the algorithm receives the positions in two dimensions of 20 tracked body points, and must to infer the correct depth of each point. We rely on a training set of 3D human motions to determine which reconstructions are plausible. Most candidate projections are unnatural motions, if not anatomically impossible, and can be eliminated on this basis. We adopt a Bayesian framework, and use the training data to compute prior probabilities of different 3D motions.

We model plausible motions as a mixture of Gaussian probabilities in a high-dimensional space. Motion capture data gathered in a professional studio provide the training data: frame-by-frame 3D coordinates for 20 tracked body points at 20-30 frames per second. We want to model the probabilities of human motions of some short duration, long enough be

informative, but short enough to characterize probabilistically from our training data. We assembled the data into short motion elements we called *snippets* of 11 successive frames, about a third of a second. We represent each snippet from the training data as a large column vector of the 3D positions of each tracked body point in each frame of the snippet.

We then use those data to build a mixture-of-Gaussians probability density model [2]. For computational efficiency, we used a clustering approach to approximate the fitting of an EM algorithm. We use $k$-means clustering to divide the snippets into $m$ groups, each of which will be modeled by a Gaussian probability cloud. For each cluster, the matrix $M_j$ is formed, where the columns of $M_j$ are the $n_j$ individual motion snippets after subtracting the mean $\mu_j$. The singular value decomposition (SVD) gives $M_j = U_j S_j V_j^T$, where $S_j$ contains the singular values along the diagonal, and $U_j$ contains the basis vectors. (We truncate the SVD to include only the 50 largest singular values.) The cluster can be modeled by a multidimensional Gaussian with covariance $\Lambda_j = \frac{1}{n_j} U_j S_j^2 U_j^T$. The prior probability of a snippet $\vec{x}$ over all the models is a sum of the Gaussian probabilities weighted by the probability of each model.

$$P(\vec{x}) = \sum_{j=1}^{m} k\pi_j e^{-\frac{1}{2}(\vec{x}-\mu_j)^T \Lambda^{-1}(\vec{x}-\mu_j)} \tag{1}$$

Here $k$ is a normalization constant, and $\pi_j$ is the *a priori* probability of model $j$, computed as the fraction of snippets in the knowledge base that were originally placed in cluster $j$. Given this approximately derived mixture-of-factors model [6], we can compute the prior probability of any snippet.

To estimate the data term (likelihood) in Bayes' law, we assume that the 2D observations include some Gaussian noise with variance $\sigma$. Combined with the prior, the expression for the probability of a given snippet $\vec{x}$ given an observation $\vec{y}$ becomes

$$P(\vec{x}, \theta, s, \vec{v}|\vec{y}) = k' \left( e^{-\|\vec{y}-R_{\theta,s,\vec{v}}(\vec{x})\|^2/(2\sigma^2)} \right) \left( \sum_{j=1}^{m} k\pi_j e^{-\frac{1}{2}(\vec{x}-\mu_j)^T \Lambda^{-1}(\vec{x}-\mu_j)} \right) \tag{2}$$

In this equation, $R_{\theta,s,\vec{v}}(\vec{x})$ is a rendering function which maps a 3D snippet $\vec{x}$ into the image coordinate system, performing scaling $s$, rotation about the vertical axis $\theta$, and image-plane translation $\vec{v}$. We use the EM algorithm to find the probabilities of each Gaussian in the mixture and the corresponding snippet $\vec{x}$ that maximizes the probability given the observations [6]. This allows the conversion of eleven frames of 2D tracking measurements into the most probable corresponding 3D snippet. In cases where the 2D tracking is poor, the reconstruction may be improved by matching only the more reliable points in the likelihood term of Equation 2. This adds a second noise process to explain the outlier data points in the likelihood term.

To perform the full 3D reconstruction, the system first divides the 2D tracking data into snippets, which provides the $\vec{y}$ values of Eq. 2, then finds the best (MAP) 3D snippet for each of the 2D observations. The 3D snippets are stitched together, using a weighted interpolation for frames where two snippets overlap. The result is a Bayesian estimate of the subject's motion in three dimensions.

## 4  Performance

The system as a whole will track and successfully 3D reconstruct simple, short video clips with no human intervention, apart from 2D pose initialization. It is not currently reliable enough to track difficult footage for significant lengths of time. However, analysis of short clips demonstrates that the system can successfully reconstruct 3D motion from ambiguous

2D video. We evaluate the two stages of the algorithm independently at first, and then consider their operation as a system.

## 4.1 Performance of the 3D reconstruction

The 3D reconstruction stage is the heart of the system. To our knowledge, no similar 2D to 3D reconstruction technique relying on prior information has been published. ([3], developed simultaneously, also uses an inference-based approach). Our tests show that the module can restore deleted depth information that looks realistic and is close to the ground truth, at least when the knowledge base contains some examples of similar motions. This makes the 3D reconstruction stage itself an important result, which can easily be applied in conjunction with other tracking technologies.

To test the reconstruction with known ground truth, we held back some of the training data for testing. We artificially provided perfect 2D marker position data, $\vec{y}$ in Eq. 2, and tested the 3D reconstruction stage in isolation. After removing depth information from the test sequence, the sequence is reconstructed as if it had come from the 2D tracker. Sequences produced in this manner look very much like the original. They show some rigid motion error along the line of sight. An analysis of the uncertainty in the posterior probability predicts high uncertainty for the body motion mode of rigid motion parallel to the orthographic projection [10]. This slipping can be corrected by enforcing ground-contact constraints. Figure 1 shows a reconstructed running sequence corrected for rigid motion error and superimposed on the original. The missing depth information is reconstructed well, although it sometimes lags or anticipates the true motion slightly. Quantitatively, this error is a relatively small effect. After subtracting rigid motion error, the mean residual 3D errors in limb position are the same order of magnitude as the small frame-to frame changes in those positions.

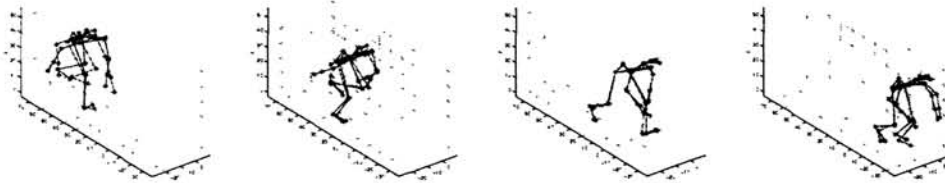

Figure 1: Original and reconstructed running sequences superimposed (frames 1, 7, 14, and 21).

## 4.2 Performance of the 2D tracker

The 2D tracker performs well under constant illumination, providing quite accurate results from frame to frame. The main problem it faces is the slow accumulation of error. On longer sequences, the errors can build up to the point where the module is no longer tracking the body parts it was intended to track. The problem is worsened by low contrast, occlusion and lighting changes. More careful body modeling [5], lighting models, and modeling of the background may address these issues. The sequences we used for testing were several seconds long and had fairly good contrast. Although adequate to demonstrate the operation of our system, the 2D tracker contains the most open research issues.

## 4.3 Overall system performance

Three example reconstructions are given, showing a range of different tracking situations. The first is a reconstruction of a stationary figure waving one arm, with most of the motion

in the image plane. The second shows a figure bringing both arms together towards the camera, resulting in a significant amount of foreshortening. The third is a reconstruction of a figure walking sideways, and includes significant self-occlusion

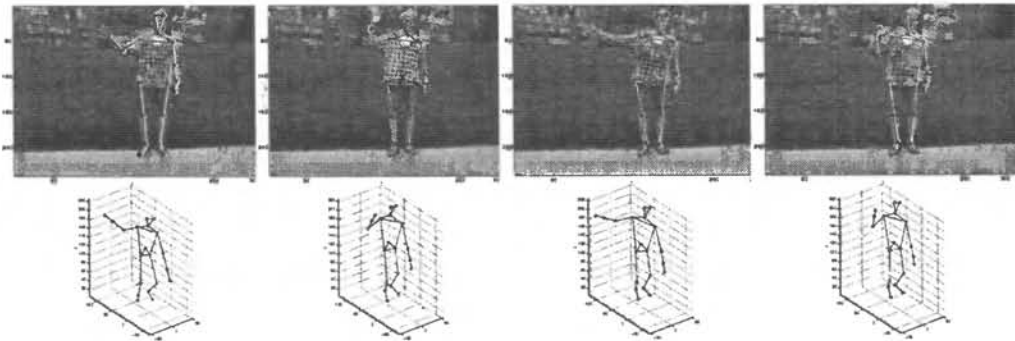

Figure 2: First clip and its reconstruction (frames 1, 25, 50, and 75).

The first video is the easiest to track because there is little or no occlusion and change in lighting. The reconstruction is good, capturing the stance and motion of the arm. There is some rigid motion error, which is corrected through ground friction constraints. The knees are slightly bent; this may be because the subject in the video has different body proportions than those represented in the training database.

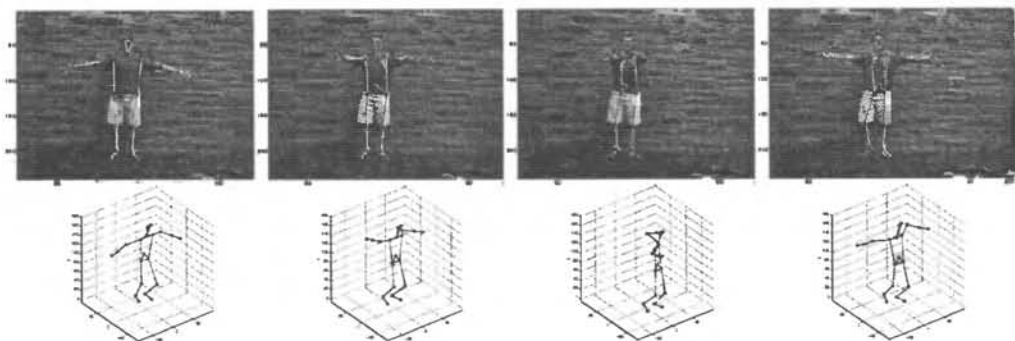

Figure 3: Second clip and its reconstruction (frames 1, 25, 50, and 75).

The second video shows a figure bringing its arms together towards the camera. The only indication of this is in the foreshortening of the limbs, yet the 3D reconstruction correctly captures this in the right arm. (Lighting changes and contrast problems cause the 2D tracker to lose the left arm partway through, confusing the reconstruction of that limb, but the right arm is tracked accurately throughout.)

The third video shows a figure walking to the right in the image plane. This clip is the hardest for the 2D tracker, due to repeated and prolonged occlusion of some body parts. The tracker loses the left arm after 15 frames due to severe occlusion, yet the remaining tracking information is still sufficient to perform an adequate reconstruction. At about frame 45, the left leg has crossed behind the right several times and is lost, at which point the reconstruction quality begins to degrade. The key to a more reliable reconstruction on this sequence is better tracking.

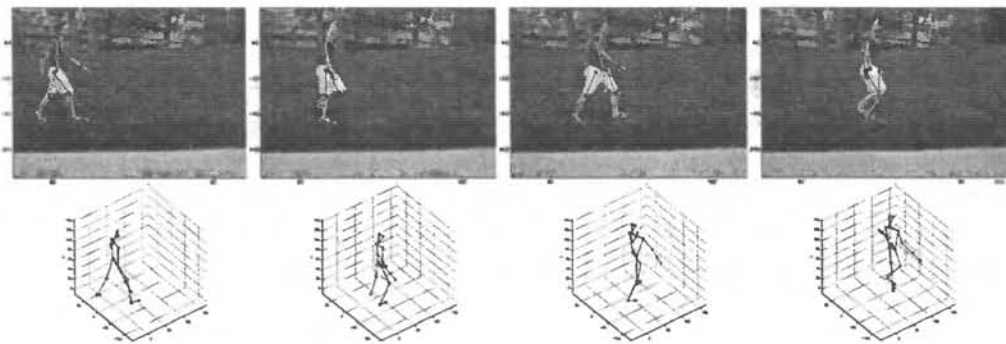

Figure 4: Third clip and its reconstruction (frames 6, 16, 26, and 36).

## 5 Conclusion

We have demonstrated a system that tracks human figures in short video sequences and reconstructs their motion in three dimensions. The tracking is unassisted, although 2D pose initialization is required. The system uses prior information learned from training data to resolve the inherent ambiguity in going from two to three dimensions, an essential step when working with a single-camera video source. To achieve this end, the system relies on prior knowledge, extracted from examples of human motion. Such a learning-based approach could be combined with more sophisticated measurement-based approaches to the tracking problem [12, 8, 4].

## References

[1] J. R. Bergen, P. Anandan, K. J. Hanna, and R. Hingorani. Hierarchical model-based motion estimation. In *European Conference on Computer Vision*, pages 237–252, 1992.

[2] C. M. Bishop. *Neural networks for pattern recognition.* Oxford, 1995.

[3] M. Brand. Shadow puppetry. In *Proc. 7th Intl. Conf. on Computer Vision*, pages 1237–1244. IEEE, 1999.

[4] C. Bregler and J. Malik. Tracking people with twists and exponential maps. In *IEEE Computer Society Conference on Computer Vision and Pattern Recognition*, Santa Barbara, 1998.

[5] D. M. Gavrila and L. S. Davis. 3d model-based tracking of humans in action: A multi-view approach. In *IEEE Computer Society Conference on Computer Vision and Pattern Recognition*, San Francisco, 1996.

[6] Z. Ghahramani and G. E. Hinton. The EM algorithm for mixtures of factor analyzers. Technical report, Department of Computer Science, University of Toronto, May 21 1996. (revised Feb. 27, 1997).

[7] L. Goncalves, E. Di Bernardo, E. Ursella, and P. Perona. Monocular tracking of the human arm in 3D. In *Proceedings of the Third International Conference on Computer Vision*, 1995.

[8] M. Isard and A. Blake. Condensation – conditional density propagation for visual tracking. *International Journal of Computer Vision*, 29(1):5–28, 1998.

[9] S. X. Ju, M. J. Black, and Y. Yacoob. Cardboard people: A parameterized model of articulated image motion. In *2nd International Conference on Automatic Face and Gesture Recognition*, 1996.

[10] M. E. Leventon and W. T. Freeman. Bayesian estimation of 3-d human motion from an image sequence. Technical Report TR98-06, Mitsubishi Electric Research Lab, 1998.

[11] D. D. Morris and J. Rehg. Singularity analysis for articulated object tracking. In *IEEE Computer Society Conference on Computer Vision and Pattern Recognition*, Santa Barbera, 1998.

[12] S. Wachter and H.-H. Nagel. Tracking of persons in monocular image sequences. In *Nonrigid and Articulated Motion Workshop*, 1997.